# Second Order Properties of Error Surfaces : Learning Time and Generalization

**Yann Le Cun**
AT&T Bell Laboratories
Crawfords Corner Rd.
Holmdel, NJ 07733, USA

**Ido Kanter**
Department of Physics
Bar Ilan University
Ramat Gan, 52100 Israel

**Sara A. Solla**
AT&T Bell Laboratories
Crawfords Corner Rd.
Holmdel, NJ 07733, USA

## Abstract

The learning time of a simple neural network model is obtained through an analytic computation of the eigenvalue spectrum for the Hessian matrix, which describes the second order properties of the cost function in the space of coupling coefficients. The form of the eigenvalue distribution suggests new techniques for accelerating the learning process, and provides a theoretical justification for the choice of centered versus biased state variables.

## 1 INTRODUCTION

Consider the class of learning algorithms which explore a space $\{W\}$ of possible couplings looking for optimal values $W^*$ for which a cost function $E(W)$ is minimal. The dynamical properties of searches based on gradient descent are controlled by the second order properties of the E(W) surface. An analytic investigation of such properties provides a characterization of the time scales involved in the relaxation to the solution $W^*$.

The discussion focuses on layered networks with no feedback, a class of architectures remarkably successful at perceptual tasks such as speech and image recognition. We derive rigorous results for the learning time of a single linear unit, and discuss their generalization to multi-layer nonlinear networks. Causes for the slowest time constants are identified, and specific prescriptions to eliminate their effect result in practical methods to accelerate convergence.

## 2   LEARNING BY GRADIENT DESCENT

Multi-layer networks are composed of model neurons interconnected through a feedforward graph. The state $x_i$ of the $i$-th neuron is computed from the states $\{x_j\}$ of the set $S_i$ of neurons that feed into it through the total input (or *induced local field*) $a_i = \sum_{j \in S_i} w_{ij} x_j$. The coefficient $w_{ij}$ of the linear combination is the coupling from neuron $j$ to neuron $i$. The local field $a_i$ determines the state $x_i$ through a nonlinear differentiable function $f$ called the *activation function* : $x_i = f(a_i)$. The activation function is often chosen to be the hyperbolic tangent or a similar sigmoid function.

The connection graph of multi-layer networks has no feedback loops, and the stable state is computed by propagating state information from the input units (which receive no input from other units) to the output units (which propagate no information to other units). The initialization of the state of the input units through an input vector $X$ results in an output vector $\mathcal{O}$ describing the state of the output units. The network thus implements an input-output map, $\mathcal{O} = \mathcal{O}(X, W)$, which depends on the values assigned to the vector $W$ of synaptic couplings.

The learning process is formulated as a search in the space $W$, so as to find an optimal configuration $W^*$ which minimizes a function $E(W)$. Given a training set of $p$ input vectors $X^\mu$ and their desired outputs $D^\mu$, $1 \le \mu \le p$, the cost function

$$E(W) = \frac{1}{2p} \sum_{\mu=1}^{p} \|D^\mu - \mathcal{O}(X^\mu, W)\|^2 \qquad (2.1)$$

measures the discrepancy between the actual behavior of the system and the desired behavior. The minimization of $E$ with respect to $W$ is usually performed through iterative updates using some form of gradient descent:

$$W(k+1) = W(k) - \eta \nabla E, \qquad (2.2)$$

where $\eta$ is used to adjust the size of the updating step, and $\nabla E$ is an estimate of the gradient of $E$ with respect to $W$. The commonly used Back-Propagation algorithm popularized by (Rumelhart, Hinton, and Williams, 1986), provides an efficient way of estimating $\nabla E$ for multi-layer networks.

The dynamical behavior of learning algorithms based on the minimization of $E(W)$ through gradient descent is controlled by the properties of the $E(W)$ surface. The goal of this work is to gain better understanding of the structure of this surface through an investigation of its second derivatives, as contained in the Hessian matrix **H**.

## 3   SECOND ORDER PROPERTIES

We now consider a simple model which can be investigated analytically: an $N$-dimensional input vector feeding onto a single output unit with a linear activation function $f(a) = a$. The output corresponding to input $X^\mu$ is given by

$$\mathcal{O}^\mu = \sum_{i=1}^{N} w_i x_i^\mu = W^T X^\mu, \qquad (3.1)$$

where $x_i^\mu$ is the $i$-th component of the $\mu$-th input vector, and $w_i$ is the coupling from the $i$-th input unit to the output.

The rule for weight updates

$$W(k+1) = W(k) - \frac{\eta}{p} \sum_{\mu=1}^{p} (\mathcal{O}^\mu - d^\mu) X^\mu \qquad (3.2)$$

follows from the gradient of the cost function

$$E(W) = \frac{1}{2p} \sum_{\mu=1}^{p} (d^\mu - \mathcal{O}^\mu)^2 = \frac{1}{2p} \sum_{\mu=1}^{p} (d^\mu - W^T X^\mu)^2. \qquad (3.3)$$

Note that the cost function of Eq. (3.3) is quadratic in $W$, and can be rewritten as

$$E(W) = \frac{1}{2}(W^T \mathbf{R} W - 2Q^T W + c), \qquad (3.4)$$

where $\mathbf{R}$ is the covariance matrix of the input, $R_{ij} = 1/p \sum_{\mu=1}^{p} x_i^\mu x_j^\mu$, a symmetric and nonnegative $N \times N$ matrix; the $N$-dimensional vector $Q$ has components $q_i = 1/p \sum_{\mu=1}^{p} d^\mu x_i^\mu$, and the constant $c = 1/p \sum_{\mu=1}^{p} (d^\mu)^2$. The gradient is given by $\nabla E = \mathbf{R} W - Q$, while the Hessian matrix of second derivatives is $\mathbf{H} = \mathbf{R}$.

The solution space of vectors $W^*$ which minimize $E(W)$ is the subspace of solutions of the linear equation $\mathbf{R} W = Q$, resulting from $\nabla E = 0$. This subspace reduces to a point if $\mathbf{R}$ is full rank. The diagonalization of $\mathbf{R}$ provides a diagonal matrix $\mathbf{\Lambda}$ formed by its eigenvalues, and a matrix $\mathbf{U}$ formed by its eigenvectors. Since $\mathbf{R}$ is nonnegative, all eigenvalues satisfy $\lambda \geq 0$.

Consider now a two-step coordinate transformation: a translation $V' = W - W^*$ provides new coordinates centered at the solution point; it is followed by a rotation $V = \mathbf{U} V' = \mathbf{U}(W - W^*)$ onto the principal axes of the error surface. In the new coordinate system

$$E(V) = \frac{1}{2} V^T \mathbf{\Lambda} V + E_0, \qquad (3.5)$$

with $\mathbf{\Lambda} = \mathbf{U}^T \mathbf{R} \mathbf{U}$ and $E_0 = E(W^*)$. Then $\partial E / \partial v_j = \lambda_j v_j$, and $\partial^2 E / \partial v_j \partial v_k = \lambda_j \delta_{jk}$. The eigenvalues of the input covariance matrix give the second derivatives of the error surface with respect to its principal axes.

In the new coordinate system the Hessian matrix is the diagonal matrix $\mathbf{\Lambda}$, and the rule for weight updates becomes a set of $N$ decoupled equations:

$$V(k+1) = V(k) - \eta \mathbf{\Lambda} V(k), \qquad (3.6)$$

The evolution of each component along a principal direction is given by

$$v_j(k) = (1 - \eta \lambda_j)^k v_j(0), \qquad (3.7)$$

so that $v_j$ will converge to zero (and thus $w_j$ to the solution $w_j^*$) provided that $0 < \eta < 2/\lambda_j$. In this regime $v_j$ decays to zero exponentially, with characteristic time $\tau_j = (\eta \lambda_j)^{-1}$. The range $1/\lambda_j < \eta < 2/\lambda_j$ corresponds to underdamped dynamics: the step size is large and convergence to the solution occurs through

oscillatory behavior. The range $0 < \eta < 1/\lambda_j$ corresponds to overdamped dynamics: the step size is small and convergence requires many iterations. Critical damping occurs for $\eta = 1/\lambda_j$; if such choice is possible, the solution is reached in one iteration (Newton's method).

If all eigenvalues are equal, $\lambda_j = \lambda$ for all $1 \leq j \leq N$, the Hessian matrix is diagonal: $\mathbf{H} = \mathbf{\Lambda}$. Convergence can be obtained in one iteration, with optimal step size $\eta = 1/\lambda$, and learning time $\tau = 1$. This highly symmetric case occurs when cross-sections of $E(W)$ are hyperspheres in the $N$-dimensional space $\{W\}$. Such high degree of symmetry is rarely encountered: correlated inputs result in nondiagonal elements for $\mathbf{H}$, and the principal directions are rotated with respect to the original coordinates. The cross-sections of $E(W)$ are elliptical, with different eigenvalues along different principal directions. Convergence requires $0 < \eta < 2/\lambda_j$ for all $1 \leq j \leq N$, thus $\eta$ must be chosen in the range $0 < \eta < 1/\lambda_{\max}$, where $\lambda_{\max}$ is the largest eigenvalue. The slowest time constant in the system is $\tau_{\max} = (\eta\lambda_{\min})^{-1}$, where $\lambda_{\min}$ is the lowest nonzero eigenvalue. The optimal step size $\eta = 1/\lambda_{\max}$ thus leads to $\tau_{\max} = \lambda_{\max}/\lambda_{\min}$ for the decay along the principal direction of smallest nonzero curvature. A distribution of eigenvalues in the range $\lambda_{\min} \leq \lambda \leq \lambda_{\max}$ results in a distribution of learning times, with average $< \tau > = \lambda_{\max} < 1/\lambda >$.

This analysis demonstrates that learning dynamics in quadratic surfaces are fully controlled by the eigenvalue distribution of the Hessian matrix. It is thus of interest to investigate such eigenvalue distribution.

## 4   EIGENVALUE SPECTRUM

The simple linear unit of Eq. (3.1) leads to the error function (3.4), for which the Hessian is given by the covariance matrix

$$R_{ij} = 1/p \sum_{\mu=1}^{p} x_i^{\mu} x_j^{\mu}. \tag{4.1}$$

It is assumed that the input components $\{x_i^{\mu}\}$ are independent, and drawn from a distribution with mean $m$ and variance $v$. The size of the training set is quantified by the ratio $\alpha = p/N$ between the number of training examples and the dimensionality of the input vector . The eigenvalue spectrum has been computed (Le Cun, Kanter, and Solla, 1990), and it exhibits three dominant features:

(a) If $p < N$, the rank of the matrix $\mathbf{R}$ is $p$. The existence of $(N-p)$ zero eigenvalues out of $N$ results in a delta function contribution of weight $(1-\alpha)$ at $\lambda = 0$ for $\alpha < 1$.

(b) A continuous part of the spectrum,

$$\rho(\lambda) = \frac{[4\alpha v^2 - (\lambda\alpha - v(1+\alpha))^2]^{1/2}}{2\pi\lambda v} \tag{4.2}$$

within the bounded interval $\lambda_- < \lambda < \lambda_+$, with $\lambda\pm = (1 \pm \sqrt{\alpha})^2 v/\alpha$ (Krogh and Hertz, 1991). Note that $\rho(\lambda)$ is controlled only by the variance $v$ of the distribution from which the inputs are drawn. The bounds $\lambda_\pm$ are well defined, and of order one. For all $\alpha < 1$, $\lambda_- > 0$, indicating a gap at the lower end of the spectrum.

(c) An isolated eigenvalue of order $N$, $\lambda_N$, present in the case of biased inputs $(m \neq 0)$.

True correlations between pairs $(x_j, x_k)$ of input components might lead to a quite different spectrum from the one described above.

The continuous part (4.2) of the eigenvalue spectrum has been computed in the $N \to \infty$ limit, while keeping $\alpha$ constant and finite. The magnitude of finite size effects has been investigated numerically for $N \leq 200$ and various values of $\alpha$. Results for $N = 200$, shown in Fig. 1, indicate that finite size effects are negligible: the distribution $\rho(\lambda)$ is bounded within the interval $[\lambda_-, \lambda_+]$, in good agreement with the theoretical prediction, even for such small systems. The result (4.2) is thus applicable in the finite $p = \alpha N$ case, an important regime given the limited availability of training data in most learning problems.

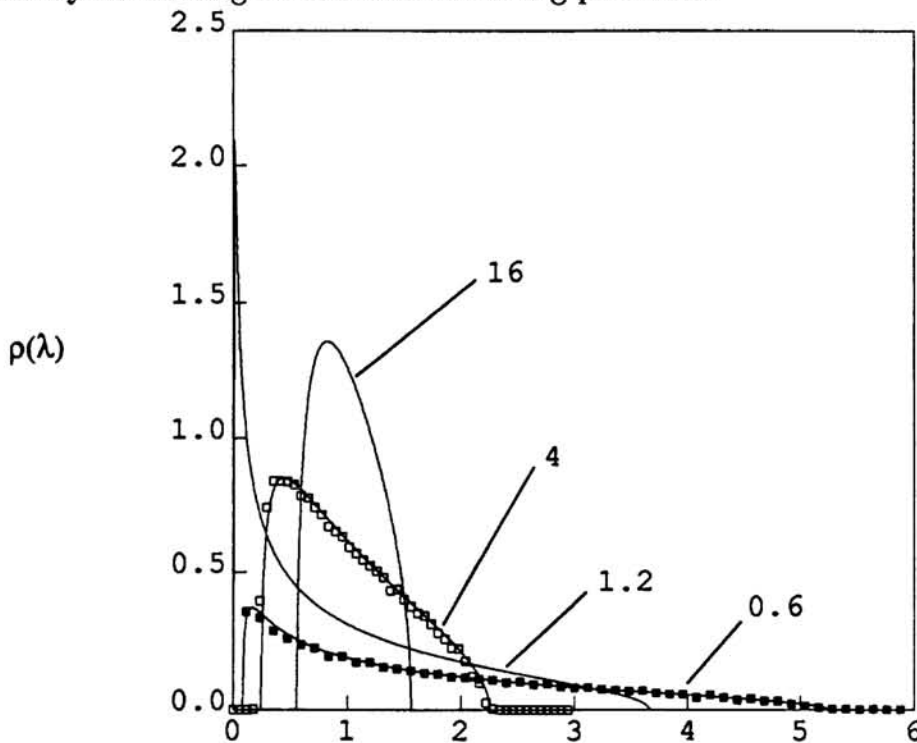

Figure 1: Spectral density $\rho(\lambda)$ predicted by Eq. (4.2) for $m = 0$, $v = 1$, and $\alpha = 0.6, 1.2, 4$, and 16. Experimental histograms for $\alpha = 0.6$ (full squares) and $\alpha = 4$ (open squares) are averages over 100 trials with $N = 200$ and $x_i^\mu = \pm 1$ with probability $1/2$ each.

The existence of a large eigenvalue $\lambda_N$ is easily understood by considering the structure of the covariance matrix $\mathbf{R}$ in the $p \to \infty$ limit, a regime for which a detailed analysis is available in the adaptive filters literature (Widrow and Stearns, 1985). In this limit, all off-diagonal elements of $\mathbf{R}$ are equal to $m^2$, and all diagonal elements are equal to $v + m^2$. The eigenvector $U_N = (1...1)$ thus corresponds to the eigenvalue $\lambda_N = Nm^2 + v$. The remaining $(N - 1)$ eigenvalues are all equal to $v$ (note that the continuous part of the spectrum collapses onto a delta function at $\lambda_- = \lambda_+ = v$ as $p \to \infty$ ), thus satisfying $\text{tr}\mathbf{R} = N(m^2 + v)$. The large part of $\lambda_N$ is eliminated for centered distributions with $m = 0$, such as $x_i^\mu = \pm 1$ with probability $1/2$, or $x_i^\mu = 3, -1, -2$ with probability $1/3$. Note that although $m$ is

crucial in controlling the existence of an isolated eigenvalue of order $N$, it plays no role in the spectral density of Eq. (4.2).

## 5 LEARNING TIME

Consider the learning time $\tau = \alpha(\lambda_{\max}/\lambda_{\min})$. The eigenvalue ratio $(\lambda_{\max}/\lambda_{\min})$ measures the maximum number of iterations, and the factor of $\alpha$ accounts for the time needed for each presentation of the full training set.

For $m = 0$, $\lambda_{\max} = \lambda_+$, and $\lambda_{\min} = \lambda_-$. The learning time $\tau = \alpha(\lambda_+/\lambda_-)$ can be easily computed using Eq. (4.2): $\tau = \alpha(1 + \sqrt{\alpha})^2/(1 - \sqrt{\alpha})^2$. As a function of $\alpha$, $\tau$ diverges at $\alpha = 1$, and, surprisingly, goes through a minimum at $\alpha = (1 + \sqrt{2})^2 = 5.83$ before diverging linearly for $\alpha \to \infty$. Numerical simulations were performed to estimate $\tau$ by counting the number $T$ of presentations of training examples needed to reach an allowed error level $\tilde{E}$ through gradient descent. If the prescribed error $\tilde{E}$ is sufficiently close to the minimum error $E_0$, $T$ is controlled by the slowest mode, and it provides a good estimate for $\tau$. Numerical results for $T$ as a function of $\alpha$, shown in Fig. 2, were obtained by training a single linear neuron on randomly generated vectors. As predicted, the curve exhibits a clear maximum at $\alpha = 1$, as well as a minimum between $\alpha = 4$ and $\alpha = 5$. The existence of such optimal training set size for fast learning is a surprising result.

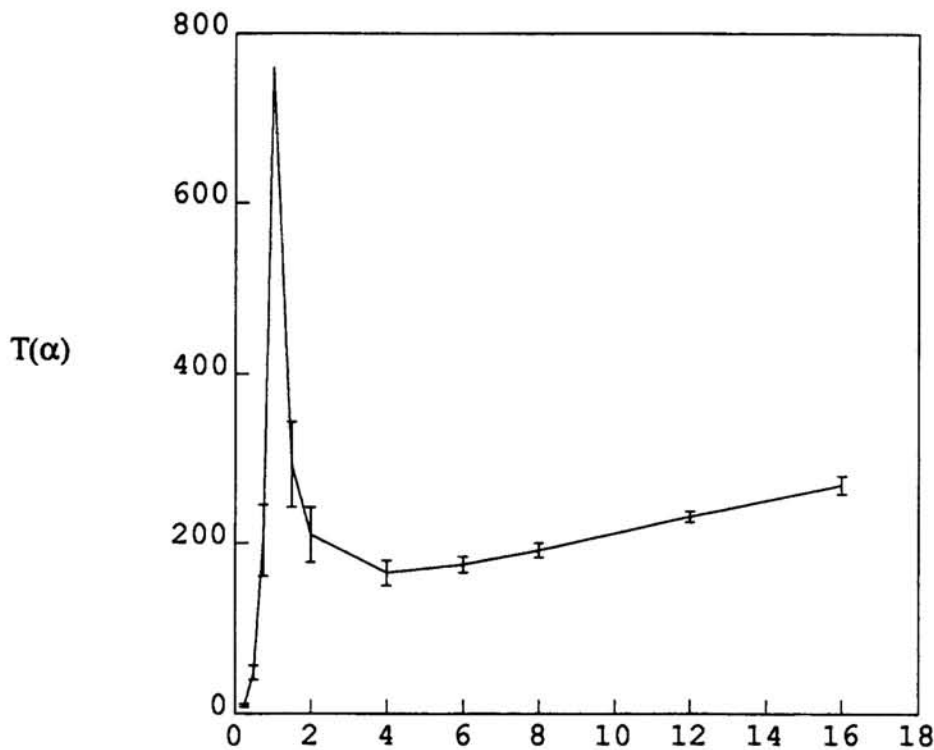

Figure 2: Number of iterations $T$ (averaged over 20 trials) needed to train a linear neuron with $N = 100$ inputs. The $x_j^\mu$ are uniformly distributed between -1 and +1. Initial and target couplings $W$ are chosen randomly from a uniform distribution within the $[-1, +1]^N$ hypercube. Gradient descent is considered complete when the error reaches the prescribed value $\tilde{E} = 0.001$ above the $E_0 = 0$ minimum value.

Biased inputs $m \neq 0$ produce a large eigenvalue $\lambda_{\max} = \lambda_N$, proportional to $N$ and responsible for slow convergence. A simple approach to reducing the learning time is to center each input variable $x_j$ by subtracting its mean. An obvious source of systematic bias $m$ is the use of activation functions which restrict the state variables to the [0,1] interval. Symmetric activation functions such as the hyperbolic tangent are empirically known to yield faster convergence than their nonsymmetric counterparts such as the logistic function. Our results provide an explanation to this observation.

The extension of these results to multi-layer networks rests on the observation that each neuron $i$ receives state information $\{x_j\}$ from the $j \in S_i$ neurons that feed into it, and can be viewed as minimizing a local objective function $E_i$ whose Hessian involves the the covariance matrix of such inputs. If all input variables are uncorrelated and have zero mean, no large eigenvalues will appear. But states with $\overline{x_j} = m \neq 0$ produce eigenvalues proportional to the number of input neurons $N_i$ in the set $S_i$, resulting in slow convergence if the connectivity is large. An empirically known solution to this problem, justified by our theoretical analysis, is to use individual learning rates $\eta_i$ inversely proportional to the number of inputs $N_i$ to the $i$-th neuron. Yet another approach is to keep a running estimate of the average $\overline{x_j}$ and use centered state variables $\tilde{x_j} = x_j - \overline{x_j}$. Such algorithm results in considerable reductions in learning time.

## 6    CONCLUSIONS

Our results are based on a rigorous calculation of the eigenvalue spectrum for a symmetric matrix constructed from the outer product of random vectors. The spectral density provides a full description of the relaxation of a single adaptive linear unit, and yields a surprising result for the optimal size of the training set in batch learning. Various aspects of the dynamics of learning in multi-layer networks composed of nonlinear units are clarified: the theory justifies known empirical methods and suggests novel approaches to reduce learning times.

### References

A. Krogh and J. A. Hertz (1991), 'Dynamics of generalization in linear perceptrons', in *Advances in Neural Information Processing Systems 3*, ed. by D. S. Touretzky and R. Lippman, Morgan Kaufmann (California).

Y. Le Cun, I. Kanter, and S. A. Solla (1990), 'Eigenvalues of covariance matrices: application to neural-network learning', *Phys. Rev.*, to be published.

D. E. Rumelhart, G. E. Hinton, and R. J. Williams (1986), 'Learning representations by back-propagating errors', *Nature* **323**, 533-536.

B. Widrow and S. D. Stearns (1985), *Adaptive Signal Processing*, Prentice-Hall (New Jersey).